# FM-Delta: Lossless Compression for Storing Massive Fine-tuned Foundation Models

**Wanyi Ning**[1]    **Jingyu Wang**[12*]   **Qi Qi**[1*]   **Mengde Zhu**[1]    **Haifeng Sun**[1]
**Daixuan Cheng**[1]    **Jianxin Liao**[1]    **Ce Zhang**[3]

[1] Beijing University of Posts and Telecommunications
[2] Pengcheng Laboratory    [3] University of Chicago
{ningwanyi, wangjingyu, qiqi8266, arnoldzhu, hfsun}@bupt.edu.cn
daixuancheng6@gmail.com, liaojx@bupt.edu.cn, cez@uchicago.edu

## Abstract

Pre-trained foundation models, particularly large language models, have achieved remarkable success and led to massive fine-tuned variants. These models are commonly fine-tuned locally and then uploaded by users to cloud platforms such as HuggingFace for secure storage. However, the huge model number and their billion-level parameters impose heavy storage overhead for cloud with limited resources. Our empirical and theoretical analysis reveals that most fine-tuned models in cloud have a small difference (delta) from their pre-trained models. To this end, we propose a novel lossless compression scheme `FM-Delta` specifically for storing massive fine-tuned models in cloud. `FM-Delta` maps fine-tuned and pre-trained model parameters into integers with the same bits, and entropy codes their integer delta. In this way, cloud only needs to store one uncompressed pre-trained model and other compressed fine-tuned models. Extensive experiments have demonstrated that `FM-Delta` efficiently reduces cloud storage consumption for massive fine-tuned models by an average of around 50% with only negligible additional time in most end-to-end cases. For example, on up to 10 fine-tuned models in the GPT-NeoX-20B family, `FM-Delta` reduces the original storage requirement from 423GB to 205GB, significantly saving cloud storage costs.

## 1   Introduction

The widespread success of pre-trained foundational models, particularly large language models(LLM), has led to the proliferation of fine-tuning(1; 2; 3; 4). An increasing number of end-users download pre-trained models from cloud platforms such as HuggingFace(5), and fine-tune them using their local relevant data. After fine-tuning, these fine-tuned models are usually uploaded to cloud storage by users for future use. This trend imposes a heavy storage overhead in the cloud. From Figure 1, we can see that the total number of models stored on HuggingFace has rapidly increased from 33,187 in 2022 to 574,270 in 2024, reflecting the prominent storage overhead on cloud providers. Among these models, fine-tuned models account for a large proportion, which exist in two forms: full fine-tuned models and parameter-efficient fine-tuned (PEFT) models(6; 7). PEFT model is generally a subset network of the original model and has much fewer parameters, so they do not exert undue strain on cloud storage. By contrast, the full fine-tuned models which have the original size place a substantial storage burden on the cloud.

We collect the fine-tuning statistical information in HuggingFace for six popular model families in Table 1, which shows the numbers of full fine-tuned and PEFT models for a certain pre-trained model. We also present the proportion of "inactive" full fine-tuned models with less than 10 monthly

---

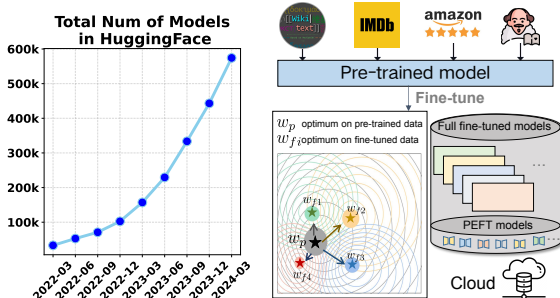

Figure 1: Pre-trained models are fine-tuned into thousands of model variants and stored in cloud.

Table 1: Fine-tuning statistical information in HuggingFace for the six most popular models on different tasks. "Inactive" refers to models with less than 10 monthly downloads.

| Model | Model size | Full num. | PEFT num. | Inact. |
|---|---|---|---|---|
| Falcon-40B | 40B | 79 | 48 | 82% |
| GPT-NeoX | 20B | 51 | 22 | 84% |
| GPT-J | 6B | 284 | 75 | 88% |
| LLaMA-7B | 7B | 5112 | 1170 | 91% |
| Bert-large | 336M | 260 | 159 | 88% |
| Stable Diff. | 860M | 1606 | 65 | 64% |
| Approx. disk storage | | 159TB | 4TB | 89% |

downloads. The approximate disk storage consumption is computed under a loose assumption that PEFT models hold 10% trainable parameters. It can be seen that although users tend to adopt PEFT methods for larger models, the key pain point in cloud storage lies in the storage overhead for full fine-tuned models. And 89% of these models are inactive. If cloud providers use the most popular SSD, SAMSUNG 870, for storage, which costs about $60 for 1TB, storing just the full fine-tuned models in Table 1 would cost $9,540, not to mention the additional costs for disk management. It is obvious that storing all the models in HuggingFace is definitely a huge expense. With the further development of LLMs, this issue will become increasingly prominent. Therefore, how to reduce the storage space of those massive fine-tuned models becomes a significant challenge for cloud providers.

The fundamental premise of reducing cloud storage costs is to safely protect the intellectual property of users (i.e., without any model alteration), since cloud providers are accountable for preserving the data integrity of all the users(8; 9). Therefore, lossless compression is a natural solution which perfectly reconstructs the original data(10; 11; 12; 13; 14). Lossless compression has been widely used to reduce the storage size of images, sound, and text, which often contain redundant data such as transparent backgrounds in images. However, we observe that directly applying traditional lossless compression techniques on models has almost no compression effect, since model itself seems to have little redundant bytes such as high element similarity suitable for compression. To apply lossless compression to massive fine-tuned models in cloud, it is desired to figure out the characteristics of those fine-tuned models and design an efficient lossless compression scheme specifically for fine-tuned models.

In this paper, we aim at mine the relationship among massive fine-tuned models in cloud and mitigate the heavy storage overhead through lossless compression. We primarily focus on compressing full fine-tuned models in cloud since PEFT models are inherently lightweight. Despite of little redundancy of model itself, we find that most fine-tuned models in cloud are much similar with their pre-trained models. Then by further analyzing the model difference theoretically, we reveal that the difference (delta) between fine-tuned and pre-trained models grows with a slow speed $\mathcal{O}(T^{\frac{1}{4}})$ as the fine-tuning steps $T$ increase. This slow growth speed results in most fine-tuned models not differing significantly from their pre-trained models, since fine-tuning is usually conducted on a similar data domain with moderate iteration steps.

Motivated by the finding, we propose a novel lossless compression scheme `FM-Delta`[2] to compress the bit-redundant delta between fine-tuned and pre-trained models. Specifically, `FM-Delta` maps the float parameters of the two models into integers with the same bits and then entropy codes the integer delta. In this way, for a certain family, the cloud only needs to hold a complete pre-trained model and other compressed fine-tuned variants, significantly reducing the storage consumption of massive full fine-tuned models. We conduct extensive experiments to demonstrate that `FM-Delta` can significantly reduce the cloud storage consumption by approximately 50% for massive full fine-tuned models, achieving a compression throughput of around 109MB/s. In our implementation, model compression and decompression processes are parallel with network transfer. This ensures that the total end-to-end time for users with a network bandwidth below 800Mbps remains nearly unchanged, with only a negligible increase in pre-trained model loading time. The 800Mbps bandwidth threshold corresponds to the compression and decompression throughput. Actually, it is the common case in the real world

considering the latest global average internet bandwidth 80Mbps(15). If compression is performed on the user side, the total time for model downloading or uploading can even be significantly reduced to as low as 70% of the original time. In addition, we give a approximate cost analysis of storage and computation in cloud, concluding that the total cost savings of the cloud can be at least 40%.

To the best of our knowledge, we are the first to reduce the cloud storage consumption of massive fine-tuned models through lossless compression. With the further explosion of fine-tuned models, the demand for efficient model storage in cloud will definitely become more prominent in the future. We hope that our work will stimulate further research in this area, efficiently storing the massive large fine-tuned models with much less cost for cloud.

## 2  Related Work

**Lossless Compression.**  Traditional lossless compression schemes mainly include Huffman coding(16), run-length coding(17), entropy coding(16) and LZ77(18). Based on these traditional schemes, researchers continuously propose new lossless compression algorithms to improve the compression rate and speed. Here we introduce five common algorithms, including LZMA(10), Gzip(12), Zlib(11), Bzip2(13), and FPzip(14). LZMA, known for its high compression rate, combines LZ77 with context modeling techniques. Gzip is a combination of LZ77 and Huffman coding, widely used in network protocols and Unix-based systems. Zlib is based on the DEFLATE algorithm(19), which also combines LZ77 and Huffman coding. Bzip2 is based on the Burrows-Wheeler transform(20) and Huffman coding. The above four algorithms compress byte streams directly without considering the data format. FPzip specifically targets scientific data and compresses the delta of adjacent elements through Lorenzo predictor(21), achieving effective compression for floating-point arrays. However, when compressing models, these lossless compression algorithms have minimal effectiveness. The existing works on applying lossless compression to models have mostly focused on compressed models that have already been quantized or sparsified(22; 23). Although Hershcovitch et al. concurrently propose a byte grouping method and apply the stardard compressors to compress float models, its compression rate is limited without reporting the actual compression speed(24). In contrast, `FM-Delta` is the first lossless compression algorithm specifically for fine-tuned models, achieving significant storage reduction in cloud.

**Delta Compression.**  Delta compression is a popular compression technique which encodes a target file relative to one or more reference files through exploiting the high redundancy between them(14; 25; 26; 27; 28; 29; 30; 31). It has been also used in the context of neural networks to compress the model updates and has demonstrated potential in accelerating model synchronization in distributed training(27; 28), accelerating multi-model serving(31; 32), and reducing the transmission and storage of model checkpoints or versions(29; 30). In the latter case which also considers storage, models are continually updated on the same data domain, with the primary focus on meeting the accuracy requirements of the final model checkpoint. Therefore, these researches use lossy techniques like quantization to compress delta between two checkpoints. In contrast, cloud storage service for massive fine-tuned models is a totally novel issue, where storage requires the data of users "exactly lossless". Besides, it is still yet to be mined about the numerical relationship between fine-tuned and pre-trained models which have different data domains. Therefore, we investigate delta between fine-tuned and pre-trained models in detail and apply lossless compression.

## 3  Difference between Fine-tuned and Pre-trained Models

### 3.1  Empirical Results

We download four common model families for different learning tasks from the popular cloud provider HuggingFace, including Stable Diffusion(33), GPT2(34), Bert-large-uncased(35), and ResNet50(36). Firstly, we measure the average cosine similarity between the fine-tuned and pre-trained models and show the results in Figure 2(a). It can be seen the similarity on all four families is higher than 0.98, which reflects that the fine-tuned model is much similar with the pre-trained model after fine-tuning. Furthermore, we show the distribution of the weight difference between fine-tuned and pre-trained models in Figure 2(b) on Pokemon Stable Diffusion(37), Wikitext103 GPT2(38), SST2 BERT(39), and FER2013 ResNet50(40). It can be seen that all the element differences are less than 1 and on SST2 Bert-large-uncased even less that 0.01. In Figure 2(c), we show the residual matrix of different layers on Wikitext103 GPT2. The element of the residual is basically an order of magnitude below $10^{-2}$. All the above observations confirm that there is usually a small difference between most

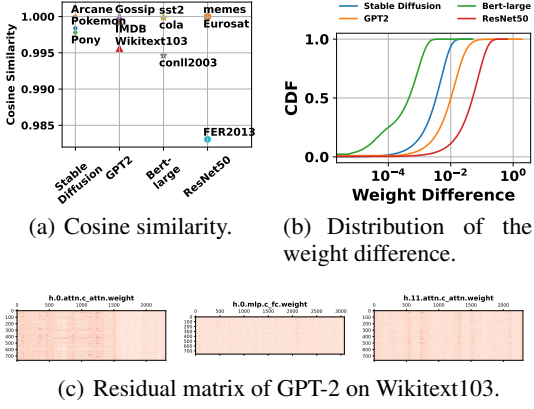

(a) Cosine similarity.

(b) Distribution of the weight difference.

(c) Residual matrix of GPT-2 on Wikitext103.

Figure 2: Difference information between the fine-tuned and pre-trained models.

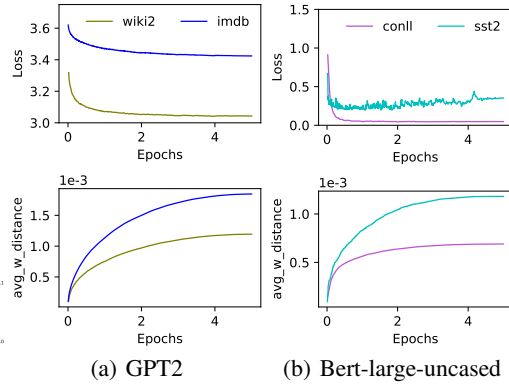

(a) GPT2

(b) Bert-large-uncased

Figure 3: Fine-tuning different models.

fine-tuned and pre-trained models stored in cloud. For further investigation, we have fine-tuned two models on different datasets for each as shown in Figure 3. We measure the average parameter element difference $\bar{\Delta}(\mathbf{w}_p, \mathbf{w}_f)$ between these two models as the training processes as follows:

$$\bar{\Delta}(\mathbf{w}_p, \mathbf{w}_f) = \frac{1}{L} \sum_{j=1}^{L} \frac{1}{d_j} \sum_{i=1}^{d_j} \left| w_{p,i}^j - w_{f,i}^j \right|, \tag{1}$$

where $d_j$ is the number of elements in the $j^{th}$ layer and $L$ is the number of layers. It can be seen that the difference grows with a slow speed as fine-tuning processes. Since most fine-tuning tasks use a small learning rate and a limited number of steps, the difference between most fine-tuned and pre-trained models in the cloud is relatively small. We present more empirical results in Appendix B.

### 3.2 Theoretical Analysis

Based on the above empirical observations, we next present our theoretical analysis under the following common assumptions(41; 42), to further understand the difference between fine-tuned and pre-trained model parameters.

*Assumption 1.* For the loss function $f$, there exists $\mathbf{w}^* \in \mathbb{R}^d$ such that $f(\mathbf{w}) \geq f(\mathbf{w}^*)$, for all $\mathbf{w}$.

*Assumption 2.* $f$ satisfies that for all $\mathbf{w}, \mathbf{v} \in \mathbb{R}^d$, $f(\mathbf{w}) - f(\mathbf{v}) \leq (\mathbf{w} - \mathbf{v})^T \nabla f(\mathbf{v}) + \frac{\beta}{2} \|\mathbf{w} - \mathbf{v}\|^2$.

*Assumption 3.* Given a data distribution $\mathcal{D}$, the variance of stochastic gradient is bounded: $\mathbb{E}_{\xi \sim \mathcal{D}} \|G(\mathbf{w}; \xi) - \nabla f(\mathbf{w})\|^2 \leq \sigma^2$.

**Theorem 1** (Growth Rate for Model Difference.). *Let $\mathbf{w}_p$ and $\mathbf{w}_f$ are the parameters of the pre-trained and fine-tuned models, respectively. The fine-tuning stage involves $T$ training steps. With learning rate $\eta_t = \frac{1}{\beta\sqrt{t}}$, $t = 1, 2, ..., T$, the distance between $\mathbf{w}_p$ and $\mathbf{w}_f$ is*

$$\mathbb{E}\left[||\mathbf{w}_f - \mathbf{w}_p||\right] \leq \frac{\sqrt{3}\sigma}{\beta} + C_1 (\ln T)^{\frac{1}{2}} + C_2 T^{\frac{1}{4}}. \tag{2}$$

*where $|| \cdot ||$ is $l_2$-norm; $f$ is the $\beta$-smooth convex loss function on the fine-tuning dataset; $\mathbf{w}^*$ is the optimal model parameter on the fine-tuning task; $C_1$ and $C_2$ are the constants related to the pre-trained model, which are $C_1 = \left( \frac{9\sigma^2}{4\beta^2} + \frac{f(\mathbf{w}_p) - f(\mathbf{w}^*)}{2\beta} \right)^{\frac{1}{2}}$ and $C_2 = \left( \frac{\sigma^2}{\beta^2} + \frac{2(f(\mathbf{w}_p) - f(\mathbf{w}^*))}{\beta} \right)^{\frac{1}{2}}$*

We present the full proof of Theorem 1 in Appendix C.1, whereas here we present the key points. We start from $\beta$-smoothness of $f$ to connect the gradient bound $||\nabla f(\mathbf{w})||^2$ with the loss function $f(\mathbf{w})$ as $||\nabla f(\mathbf{w})||^2 \leq \frac{2}{2\eta - \beta\eta^2} (f(\mathbf{w}) - f(\mathbf{w}^*)) + \frac{\beta\eta\sigma^2}{2 - \beta\eta}$, where the learning rate satisfies $0 < \eta < \frac{2}{\beta}$. Furthermore, we utilize the model optimization step $\mathbf{w}_f = \mathbf{w}_p - \sum_{t=0}^{T} \eta_t G(\mathbf{w}_t)$ in stochastic gradient descent (SGD)(43) to connect the model weight distance with the stochastic gradient. With learning rate $\eta_t = \frac{1}{\beta\sqrt{t}}$, $t = 1, 2, ..., T$, we have,

Table 2: Comparison of a certain element value in the $i^{th}$ position of the pre-trained model ($w_p$) and the fine-tuned model ($w_f$) respectively. The delta of the two original element bytes contains a large number of redundant "0" bits.

|        |        | byte number | 1 2 3 4 |
|--------|--------|-------------|---------|
| $w_f$ | 0.0316 | int($w_p$) | 3d 01 6f 00 |
| $w_p$ | 0.0309 | int($w_f$) | 3c fd 21 ff |
| $w_f - w_p$ | 0.0007 | int($w_f - w_p$) | 3a 37 80 34 |
|        |        | int($w_f$)-int($w_p$) | 00 04 4d 01 |

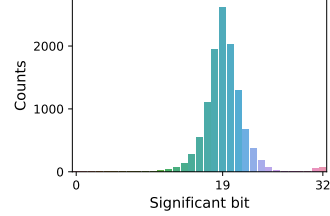

Figure 4: Most significant bit distribution of the first convolutional-layer delta.

$$\mathbb{E}\left[||\mathbf{w}_f - \mathbf{w}_p||^2\right] \leq \frac{3\sigma^2}{\beta^2} + \frac{\sigma^2}{2\beta^2}\left(4\ln(T) + 2\sqrt{T} + \frac{1}{2}\ln(T)\right) + \frac{2\sqrt{T} + \frac{1}{2}\ln(T)}{\beta}(f(\mathbf{w}_p) - f(\mathbf{w}^*)). \tag{3}$$

Rearranging the above inequality and taking the square root, we derive Theorem 1, which demonstrates the growth speed $\mathcal{O}(T^{\frac{1}{4}})$ of model difference $||\mathbf{w}_f - \mathbf{w}_p||$ with the number of fine-tuning steps $T$, which is consistent with our empirical results in Figure 3.

If we further have the assumptions of domain adaptation(44), we can relate the model difference to the two data distributions of the pre-trained $\mathcal{T}_P$ and fine-tuned $\mathcal{T}_F$ domains as follows:

$$f_{\mathcal{T}_F}(\mathbf{w}_p) \leq \hat{f}_{\mathcal{T}_P}(\mathbf{w}_p) + \sqrt{\frac{4}{m}\left(d\log\frac{2em}{d} + \log\frac{4}{\delta}\right)} + d_{\mathcal{H}\Delta\mathcal{H}}\left(\mathcal{D}_P, \mathcal{D}_F\right) + \lambda, \tag{4}$$

where $\hat{f}_{\mathcal{T}_P}(\mathbf{w}_p)$ is the empirical loss of the pre-trained model $\mathbf{w}_p$ on the pre-trained dataset with $m$ samples; $f_{\mathcal{T}_F}(\mathbf{w}_p)$ is equivalent to $f(\mathbf{w}_p)$ in $C_1$ and $C_2$ of Theorem 1. We provide more theoretical details in Appendix C.2. It can be seen that $C_1$ and $C_2$ increase monotonically with $f_{\mathcal{T}_F}(\mathbf{w}_p)$. Therefore, we conclude that there are two key factors affecting the model distance, including (1) the number of fine-tuning steps $T$, and (2) the data distribution divergence between the fine-tuned and pre-trained domains $d_{\mathcal{H}\triangle\mathcal{H}}(\mathcal{D}_F, \mathcal{D}_P)$. Generally, the data distribution divergence between the two domains exists but is not huge(45; 46). Besides, the number of fine-tuning steps $T$ is also not large in many downstream tasks, leading to a small difference between most fine-tuned and pre-trained models as in our empirical finding. This finding motivates `FM-Delta` to compress such a difference for storing massive fine-tuned models.

## 4 FM-Delta

### 4.1 Algorithm

**Mapping Float into Integer for Delta.** Given a full model pair <pre-trained model $\mathcal{M}_p$, fine-tuned model $\mathcal{M}_f$>, the key idea of `FM-Delta` is to losslessly compress their delta, since most fine-tuned models stored in cloud are similar with their pre-trained models as analyzed in Section 3. However, directly performing parameter-level subtraction will result in a float delta that lacks bit-redundancy and could even lead to lossiness(47; 14). As shown in Table 2, floating-point subtraction does not yield bit-redundant delta, while the bit-level integer subtraction yields the integer delta with a lot of "0" bits. Therefore, we determine to regard the mapped integer delta as the compression object.

As in (14), we firstly keep the same bit stream to convert the original floating-point parameters of the fine-tuned and pre-trained models to signed integers. Furthermore, to deal with complement arithmetic implemented on most platforms, we secondly map the signed integer to the unsigned integer, by flipping the most significant bit for positive floats and flipping all bits for negative floats to scale the signed range $[-2^{31}, 2^{31})$ with 32 bits to the unsigned range $[0, 2^{32})$, as illustrated in the left half of Figure 5. This transformation monotonically maps floats to unsigned integers, maintaining both the order and the linearity of differences for floats with the same sign and exponent. Finally, we conduct the unsigned integer subtraction to obtain our compression object – *the bit-redundant integer delta.* Figure 4 shows the most significant bit distribution of delta in the most popular fine-tuned Stable Diffusion model(37) in HuggingFace. The most significant bit is the bit in a binary number

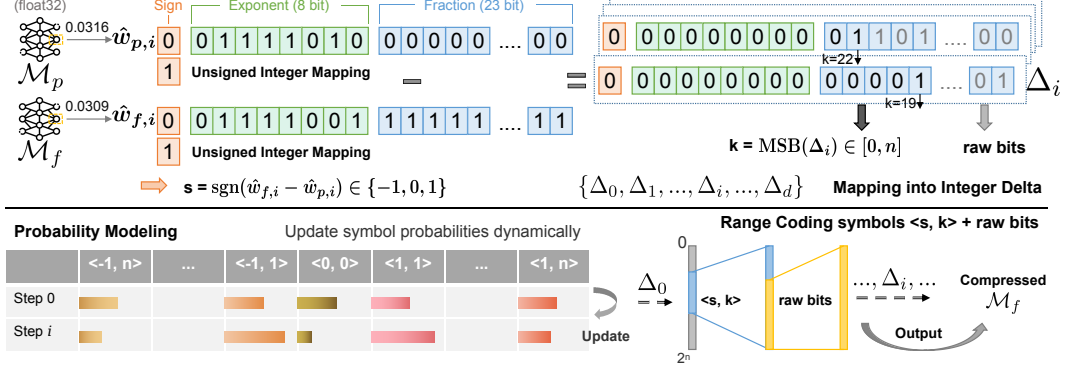

Figure 5: The lossless compression workflow of `FM-Delta`. The `FM-Delta` scheme (1) maps the two floating-point parameter elements at the same position of fine-tuned and pre-trained models into unsigned integers, and performs integer subtraction to obtain the bit-redundant delta element. Then it (2) regards the sign $s$ and the most significant bit $k$ of delta as symbols. With a quasi-static probability modeler, it encodes the symbols and scales the range to involve raw bits on all delta elements, leading to the compressed fine-tuned model.

that has the highest value position. It can be seen that the most significant bits of most delta elements are around 19, which is significantly below the full 32-bit representation, imbuing potential for further compression.

**Compression with Range Coding.** Since each delta element in the array is represented by a fixed 32 or 16 bit number and the redundant "0" bits appear in the high position, we determine to entropy code the most significant bit with range coding(48) and as in (14) to store the range coded symbols and the remaining raw bits in the lower position as the compressed fine-tuned model delta. As with most range coding processes, our compression algorithm mainly consists of the following steps. Firstly, we set an initial range interval and use the quasistatic probability modeler in (48) for probability modeling. Secondly, we define the $2n + 1$ symbols by grouping the delta elements into a small set of intervals, where $n$ is the number of bits. The $i^{th}$ integer delta element can be represented as:

$$\hat{\Delta}_i = \hat{w}_{f,i} - \hat{w}_{p,i} = s\left(2^k + m\right) \quad \in [-2^n, 2^n) \tag{5}$$

where $s$ encodes the sign of $\hat{\Delta}_i$, $0 \le k \le n$ is the position of the most significant bit, and $m$ is the remaining raw bit stream. We symbolize $g = s(k + 1)$ for the range coding. Initially, each symbol is assigned an equal frequency. As we encode the data, the probability modeler will update the symbol frequencies dynamically based on the processed data. It should be noted that to avoid running out of finite initial range $[l, l + r)$, the most significant byte will be periodically output so as to scale the range. Thirdly, we determine how to encode the whole delta matrix. In general, except for the most significant bit that can be used as symbols, the remaining raw bits of delta elements do not have much regularity for further compression(14). Therefore, the compressed bytes include both the encoded symbol interval bytes through range coding and the remaining raw bits of the delta.

Similarly for the decoding procedure, with the same equal-frequency initialization, range coding maps the encoded model parameters back to the original symbol range and termly updates the probability model. Then we get the original float-point fine-tuned model through reverse-mapping delta. Up to this point, the design of `FM-Delta` is complete. We present its workflow in Figure 5 which illustrates the key points discussed above.

## 4.2 Robustness to Difference Range

To further assess the robustness of `FM-Delta` across a wide range of differences between fine-tuned and pre-trained model parameters, we present the following theorem, which relates the most significant bit position $r$ of the integer delta to the original model parameters.

**Theorem 2** (Bit-Redundancy of Delta). *Given a specific floating-point encoding format, it has $n_s$ bits for the sign part, $n_e$ bits for the exponent part, and $n_m$ bits for the fraction part. Let $w_f = (-1)^{s_f} \times 2^{e_f} \times m_f$ and $w_p = (-1)^{s_p} \times 2^{e_p} \times m_p$ are the floating-point parameter elements*

Table 3: Given a base value 0.001, the most significant bit position $r$ of the integer delta, corresponding to the range intervals of different tuned values.

| Value | (0.001, 0.002) | [0.002, 0.004) | [0.004, 0.016) | [0.016, 0.256) | [0.256, 65.536) |
|---|---|---|---|---|---|
| r | (1, 23) | 24 | 25 | 26 | 27 |

*in the same position of the fine-tuned and pre-trained models, respectively. Let $\hat{w}_f$ and $\hat{w}_p$ are the unsigned integers mapped from $w_f$. Assuming that $w_f > w_p$, let $r$ is the most significant bit position of the integer delta, we have*

$$
\begin{aligned}
r &= \left\lceil \log_2 \left( 2^{(n_e+n_m)}(s_f \oplus s_p) + 2^{n_m}(e_f - e_p) + \hat{m}_f - \hat{m}_p \right) \right\rceil \\
&\leq \begin{cases} \left\lceil n_m + \log_2 \left( \log_2 \left( \dfrac{w_f}{w_p} \right) + 2 \right) \right\rceil, & s_f \oplus s_p = 0 \\ n_s + n_e + n_m, & s_f \oplus s_p = 1 \end{cases}
\end{aligned}
\tag{6}
$$

*where $\lceil \cdot \rceil$ is the ceiling function; $\oplus$ is the XOR function; $\hat{m}_f$ and $\hat{m}_p$ are the integer values with the same bit streams as $m_f$ and $m_p$.*

The full proof of Theorem 2 is provided in Appendix C.3, which is derived based on the encoding standards for floats and integers. The impact on $r$ can be observed to diminish successively in sign, exponent, and fraction. Taking the 32-bit floating-point number as an example, following the most common IEEE 754 standard(49), there are respectively 1, 8, 23 bits in the sign, exponent, and fractional parts (e.g. $n_s = 1$, $n_e = 8$, and $n_m = 23$). According to Inequality (6), with the same sign, the upper bound for the most significant bit position of the integer delta $r$ is 25 when $\frac{w_f}{w_p} = 2^2$, is 26 when $\frac{w_f}{w_p} = 2^6$, and is 27 when $\frac{w_f}{w_p} = 2^{14}$, which is rare as experimentally and theoretically analyzed in Section 3. From $\frac{w_f}{w_p} = 2^2$ to $\frac{w_f}{w_p} = 2^{14}$, $w_f$ differs from $w_p$ with numerically $2^{12} \times$ scaling, while the upper bound bit $r$ for delta only increases two bits. For the sake of clarity, we further illustrate in Table 3 the most significant bit position $r$ corresponding to different ranges of fine-tuned element values, given a base (i.e. pre-trained) element value of 0.001. It can be observed that `FM-Delta` can accommodate a vast range of difference, with the most significant bit increasing by only 4 bits even in an extreme case where the value changes from 0.001 to 65.536. Therefore, we deduce that although `FM-Delta` benefits more with small tuning steps which has a higher similarity between models, it nonetheless exhibits considerable robustness for a large range of model differences.

## 5  Experiments

**Setup.** We implemented `FM-Delta` for models in PyTorch(50) format. Our experiments were run on AMD Ryzen 9 5950X 16-Core Processors@2.2GHz (32 logical processors) with 251GB of main memory. In our end-to-end simulation, we simulate the communication between cloud and users through Python `"socket"` library. Model compression and decompression processes are parallel with network transfer through reading and writing models in chunks. We download seven popular model families in HuggingFace, including Falcon-40B(51), GPT-NeoX-20B(52), GPT-J(53), GPT-2(54), Bert-large-uncased(35), Stable-Diffusion(33), and ResNet50(36). We collect a varying number of their fine-tuned models ranging from 5 to 100, involving various downstream tasks. We primarily select the fine-tuned variants based on download counts, since the more popular model is intuitively more likely to ensure quality. We compare `FM-Delta` with the five state-of-the-art lossless compression schemes, including LZMA(10), GZip(12), Zlib(11), FPZip(14), and BZip2(13). We apply these schemes on the fine-tuned model parameters to show their compression rates.

**Overall Compression Rates.** We report the overall compression rate $r$ under the different number of fine-tuned models in Table 4, which is obtained as $r = \frac{(1+n)\cdot\mathcal{M}}{\mathcal{M}+\sum_{i=0}^{n}\mathbf{Q}(\mathcal{M}_i)}$, where $n$ is the number of fine-tuned models, $\mathcal{M}$ is the pre-trained model size, and $\mathbf{Q}(\mathcal{M}_i)$ is the $i^{th}$ compressed fine-tuned model size. It can be seen that the traditional lossless compression schemes applied directly to model bytes have little compression effect with a compression rate about 90%. Surprisingly, `FM-Delta` significantly reduces storage consumption by approximately 50% to 60% of the original. Besides, the more fine-tuned models are stored, the more pronounced the compression effect becomes, since the proportion of storage occupied by pre-trained models correspondingly decreases.

Table 4: Overall compression rates and throughput of six lossless compression schemes on different model families.

| Family | Pretrained Size | Finetuned Num. | Original Storage (GB) | Storage after Compression (GB) | | | | | |
|---|---|---|---|---|---|---|---|---|---|
| | | | | LZMA | Gzip | Zlib | FPZip | BZip2 | FM-Delta |
| Falcon-40B (fp16) | 40B | 5 | 461.6 | 349.3 | 373.4 | 373.4 | 456.9 | 342.7 | **270.8 (59%)** |
| | | 10 | 846.3 | 621.7 | 669.9 | 669.9 | 837.8 | 608.5 | **473.9 (56%)** |
| GPT-NeoX (fp16) | 20B | 5 | 230.8 | 162.9 | 177.2 | 176.4 | 213.4 | 158.6 | **112.4 (49%)** |
| | | 10 | 423.2 | 298.7 | 324.9 | 323.4 | 391.2 | 290.7 | **205.2 (48%)** |
| GPT-J (fp16) | 6B | 5 | 68.4 | 57.2 | 60.6 | 60.6 | 61.2 | 58.7 | **44.6 (65%)** |
| | | 10 | 125.3 | 104.8 | 111 | 111 | 112.2 | 107.6 | **73.8 (59%)** |
| GPT-2 | 124M | 50 | 24.2 | 21.8 | 22 | 22 | 21.9 | 22.5 | **15 (62%)** |
| | | 100 | 48 | 43.2 | 43.5 | 43.5 | 43.4 | 44.5 | **28.7 (60%)** |
| Bert-large-uncased | 336M | 50 | 63.7 | 58.6 | 59.1 | 59.1 | 58.9 | 60.4 | **41.3 (65%)** |
| | | 100 | 126.1 | 116.1 | 117.1 | 117.1 | 116.6 | 119.6 | **82.1 (65%)** |
| Stable-Diffusion UNet | 860M | 5 | 19.2 | 17.7 | 17.8 | 17.8 | 17.8 | 18.3 | **12.8 (67%)** |
| | | 10 | 35.2 | 32.5 | 32.7 | 32.7 | 32.6 | 33.5 | **23.5 (67%)** |
| ResNet50 | 26M | 10 | 1.1 | 0.9 | 0.9 | 0.9 | 0.9 | 0.9 | **0.7 (68%)** |
| | | 20 | 2 | 1.7 | 1.7 | 1.7 | 1.7 | 1.8 | **1.3 (66%)** |
| | Avg. Compression Throughput (MB/s) | | | 4.9 | 36.1 | 35.6 | 83.5 | 12.1 | **109.7** |
| | Avg. Decompression Throughput (MB/s) | | | 24.8 | 236.6 | **260.8** | 80.6 | 23.8 | 100.9 |

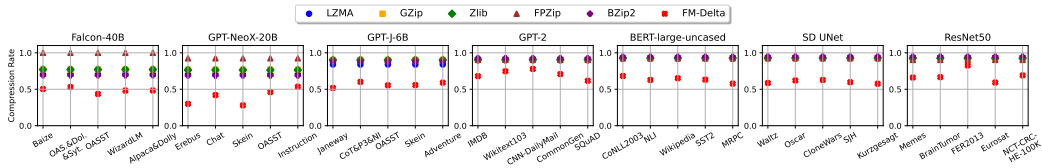

Figure 6: The single compression rates of the six lossless compression schemes on different downstream fine-tuned models.

**Single Compression Rates.** Figure 6 shows the single compression rate of the top-5 most popular fine-tuned models in each family. The single compression rate $r_i$ on the fine-tuned model $\mathcal{M}_i$ is obtained as $r_i = \mathbf{Q}(\mathcal{M}_i)/\mathcal{M}$. We can observe that directly applying traditional lossless compression algorithms to the model yields similarly marginal compression effects, irrespective of the task setting. In contrast, FM-Delta always outperforms those traditional lossless compression algorithms with a significant compression rate.

**Compression & Decompression Throughput.** We also report the throughput of compressing from memory to disk and decompressing from disk to memory in Table 4, which includes disk writing and reading process through pickling in the binary with "pickle" library. Among all the schemes, FM-Delta shows its superiority with the highest throughput 109.7MB/s in compression and the third highest throughput 100.9MB/s in decompression.

**Compression Rates of Baselines on Different Objects.** We apply the five lossless compression baselines to four different compression objects on the model pair <"bert-large-uncased", "Jorgeutd/bert-large-uncased-finetuned-ner">, including "float model parameters", "float delta", "signed int delta", and "unsigned int delta". FPzip is specifically for floating-point array, so we only show its results on float parameters and delta. Table 5 shows that delta becomes more bit-redundant after the mapping from float to unsigned integer, resulting in a better compression effect.

**Different Data Types.** Table 6 shows the compression rate of FM-Delta on Bert-large-uncased under three different data types. We can see that the compression effectiveness ranks from high to low as follows: bfloat16, float16, float32. This result is reasonable since parameter in float32 retains more fine-grained difference between the fine-tuned and pre-trained models, and bfloat16 has more exponent bits compared with float16, leading to a larger dynamic range with lower precision. It is worth mentioning that nowadays many users fine-tune models in float16 or bfloat16 to reduce computation overhead, benefits more from FM-Delta.

Table 5: Compression rates the five baselines on different objects. The compression rate of `FM-Delta` is 68%.

| Compressor | Float Params | Float Delta | Int Delta | Uint Delta |
|---|---|---|---|---|
| LZMA | 92% | 78% | 74% | 72% |
| Gzip | 92% | 86% | 83% | 82% |
| Zlib | 92% | 86% | 83% | 82% |
| FPzip | 92% | 92% | - | - |
| Bzip2 | 94% | 79% | 77% | 74% |

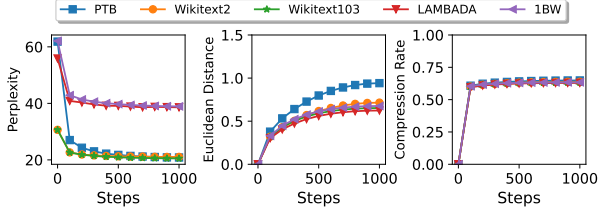

Figure 7: Three metrics over the iteration steps $T$ when fine-tuning GPT-2 on different datasets.

Table 6: Compression rates of `FM-Delta` under three different data types on Bert-large-uncased.

| Finetuned Model & Num. | FP32 | FP16 | BF16 |
|---|---|---|---|
| CoNLL2003(58) | 68% | 55% | 37% |
| NLI(59) | 63% | 45% | 27% |
| Wikipedia(60) | 65% | 49% | 32% |
| SST2(39) | 63% | 49% | 27% |
| MRPC(61) | 57% | 39% | 22% |
| 100 | 65% | 50% | 33% |

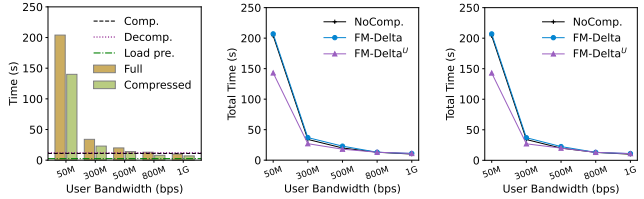

(a) Time for key procedures.  (b) Total time for upload.  (c) Total time for download.

Figure 8: End-to-end time under different user bandwidths.

**Robustness in Fine-tuning.** To observe how the compression rate of `FM-Delta` changes during fine-tuning, we fine-tune GPT-2 in float32 type on five different datasets(55; 56; 56; 34; 57). As shown in Figure 7, the Euclidean distance between the two models grows slowly as the number of fine-tuning steps increases, which is consistent with our analysis in Section 3. It can be seen that the compression rate of `FM-Delta` grows from around 60% once fine-tuning starts at an extremely slow speed with the number of steps. For example, the compression rate on PTB only increases from 60.7% to 64.8% after 1,000 fine-tuning steps. Among these datasets, the difference in Euclidean distance is obvious while the difference in compression rate is much small, up to 2%.

**Compression & Decompression in Users.** We discuss another special but common case where users have cached pre-trained models and conduct compression or decompression. We name this variant as `FM-Delta`$^U$. We explore different user bandwidths on <Bert-large-uncased(35), Jorgeutd/bert-large-uncased-finetuned-ner(58). We provide the detailed time for the key procedures in Figure 8(a), the total time for model upload and download in Figure 8(b) and Figure 8(c). It can be seen that transferring compressed model significantly decreases the transfer time from 204s to 143s under 50Mbps bandwidth. The total time with `FM-Delta` increases only by a negligible amount for pre-trained model loading when the bandwidth is less than 800Mbps, which is corresponding to the compression and decompression throughput around 100MB/s. The latest report shows that global average broadband speeds are 80Mbps for download and 35Mbps for upload, which are far below 800Mbps(15). Therefore, most users won't be significantly affected, and even those with higher speeds can tolerate the additional transfer time since model transfer isn't as speed-critical as streaming media.

Table 7: RES and total runtime under different numbers of parallel decompression requests. Each decompression task is assigned to one CPU processor.

| Parallelism | 1 | 2 | 8 | 16 | 24 |
|---|---|---|---|---|---|
| RES | 1.9GB | 3.4GB | 12.5GB | 24.1GB | 39.7GB |
| Total Time | 11.9s | 12.4s | 16.8s | 19.1s | 28.5s |

**Cloud Cost Analysis.** In the real cloud system, it is inevitable that multiple users may download or upload at the same time. We assess the total runtime of decompression with varying parallelism levels in Table 7. The result shows that the total time increases slightly with the number of parallel processes due to extra I/O operations. Although we hold that the real-world cloud has powerful elastic computing resources(62), we give the following analysis on storage and computing hardware cost for better understanding. Assuming that the server stores a model of size $M$ with $n$ fine-

tuned variants, we aim to maximize loadable models $n$ while ensuring the probability of concurrent compression/decompression tasks exceeding the server threshold is below 1%, i.e., $\sum_{k=t}^{n} P(X = k) \leq 0.01$. Regarding task concurrency as a binomial distribution, we find the maximum $n_{max} = 35,300$ with python "scipy" library. With a compression rate of 50% for FM-Delta and 89% inactive models referring to Table 1, we can get the saved storage cost $c_s$ for those inactive models is $5522. Considering our server purchase on Amazon costs $1179, if compressing inactive models from the 35300 6GB models, the total cost is saved at least $4343 ($5522 - $1179). This reduces the total cost to 60% of the original. This underscores the significant cost benefits from FM-Delta in real-world cloud scenarios. The detailed analysis is presented in Appendix E.7.

## 6 Conclusion and Limitation

In this paper, we empirically and theoretically investigate the difference delta between fine-tuned and pre-trained models. Based on our analysis of the slowly-growing delta, we propose FM-Delta, the first lossless compression scheme specifically for storing massive fine-tuned foundation models in cloud. Our experiments on up to 100 fine-tuned models demonstrate that FM-Delta can efficiently reduce cloud storage space by around 50% (e.g., from 423GB to 205GB for GPT-NeoX). This reduction incurs only negligible pre-trained model loading time in the most common case where user bandwidth is below 800Mbps. If the fine-tuned models are compressed and decompressed on the user side, the total time for model upload and download can even be reduced under low bandwidths. In terms of limitations, our method does not provide a compression support for models in GPU, as it is not necessary in our target scenario of cloud storage reduction. These are natural directions for future work to dynamically compressing massive fine-tuned models during training or inference.

## Acknowledgments and Disclosure of Funding

This work was done during Wanyi Ning's visiting study at ETH Zurich. We are sincerely grateful to everyone who supported us. This work is funded in part by the National Natural Science Foundation of China under Grants (U23B2001, 62101064, 62171057, 62201072, 62001054, 62071067), the Ministry of Education and China Mobile Joint Fund (MCM20200202, MCM20180101), and the China Scholarship Council program (Project ID: 202206470044).

## Footnotes

[2]Our code is available in https://github.com/ningwanyi/FM-Delta.

## References

[1] L. Zou, S. Zhang, H. Cai, D. Ma, S. Cheng, S. Wang, D. Shi, Z. Cheng, and D. Yin, "Pre-trained language model based ranking in baidu search," in *Proceedings of the 27th ACM SIGKDD Conference on Knowledge Discovery & Data Mining*, 2021, pp. 4014–4022.

[2] J. Dodge, G. Ilharco, R. Schwartz, A. Farhadi, H. Hajishirzi, and N. Smith, "Fine-tuning pretrained language models: Weight initializations, data orders, and early stopping," *arXiv preprint arXiv:2002.06305*, 2020.

[3] N. Ruiz, Y. Li, V. Jampani, Y. Pritch, M. Rubinstein, and K. Aberman, "Dreambooth: Fine tuning text-to-image diffusion models for subject-driven generation," *arXiv preprint arXiv:2208.12242*, 2022.

[4] M. Sun, K. Zhou, X. He, Y. Wang, and X. Wang, "Gppt: Graph pre-training and prompt tuning to generalize graph neural networks," in *Proceedings of the 28th ACM SIGKDD Conference on Knowledge Discovery and Data Mining*, 2022, pp. 1717–1727.

[5] T. Wolf, L. Debut, V. Sanh, J. Chaumond, C. Delangue, A. Moi, P. Cistac, T. Rault, R. Louf, M. Funtowicz *et al.*, "Huggingface's transformers: State-of-the-art natural language processing," *arXiv preprint arXiv:1910.03771*, 2019.

[6] S. Mangrulkar, S. Gugger, L. Debut, Y. Belkada, and S. Paul, "Peft: State-of-the-art parameter-efficient fine-tuning methods," https://github.com/huggingface/peft, 2022.

[7] E. J. Hu, Y. Shen, P. Wallis, Z. Allen-Zhu, Y. Li, S. Wang, L. Wang, and W. Chen, "Lora: Low-rank adaptation of large language models," in *The Tenth International Conference on Learning Representations, ICLR 2022, Virtual Event, April 25-29, 2022*. OpenReview.net, 2022. [Online]. Available: https://openreview.net/forum?id=nZeVKeeFYf9

[8] Y. Yang, M. Cheng, Y. Ding, and W. Zhang, "A visually meaningful image encryption scheme based on lossless compression spiht coding," *IEEE Transactions on Services Computing*, vol. 16, no. 4, pp. 2387–2401, 2023.

[9] S. Luo, G. Zhang, C. Wu, S. U. Khan, and K. Li, "Boafft: Distributed deduplication for big data storage in the cloud," *IEEE Transactions on Cloud Computing*, vol. 8, no. 4, pp. 1199–1211, 2020.

[10] E. J. Leavline and D. Singh, "Hardware implementation of lzma data compression algorithm," *International Journal of Applied Information Systems (IJAIS)*, vol. 5, no. 4, pp. 51–56, 2013.

[11] P. Deutsch and J.-L. Gailly, "Zlib compressed data format specification version 3.3," Tech. Rep., 1996.

[12] P. Deutsch, "Gzip file format specification version 4.3," Tech. Rep., 1996.

[13] J. Gilchrist, "Parallel data compression with bzip2," in *Proceedings of the 16th IASTED international conference on parallel and distributed computing and systems*, vol. 16, no. 2004. Citeseer, 2004, pp. 559–564.

[14] P. Lindstrom and M. Isenburg, "Fast and efficient compression of floating-point data," *IEEE transactions on visualization and computer graphics*, vol. 12, no. 5, pp. 1245–1250, 2006.

[15] K. MacMillan, T. Mangla, J. Saxon, N. P. Marwell, and N. Feamster, "A comparative analysis of ookla speedtest and measurement labs network diagnostic test (ndt7)," *Proceedings of the ACM on Measurement and Analysis of Computing Systems*, vol. 7, no. 1, pp. 1–26, 2023.

[16] D. E. Knuth, "Dynamic huffman coding," *Journal of algorithms*, vol. 6, no. 2, pp. 163–180, 1985.

[17] S. Golomb, "Run-length encodings (corresp.)," *IEEE transactions on information theory*, vol. 12, no. 3, pp. 399–401, 1966.

[18] S. Rigler, W. Bishop, and A. Kennings, "Fpga-based lossless data compression using huffman and lz77 algorithms," in *2007 Canadian conference on electrical and computer engineering*. IEEE, 2007, pp. 1235–1238.

[19] S. Oswal, A. Singh, and K. Kumari, "Deflate compression algorithm," *International Journal of Engineering Research and General Science*, vol. 4, no. 1, pp. 430–436, 2016.

[20] G. Manzini, "An analysis of the burrows—wheeler transform," *Journal of the ACM (JACM)*, vol. 48, no. 3, pp. 407–430, 2001.

[21] L. Ibarria, P. Lindstrom, J. Rossignac, and A. Szymczak, "Out-of-core compression and decompression of large n-dimensional scalar fields," *Computer Graphics Forum*, vol. 22, pp. 343–348, 06 2003.

[22] X. Zhang, R. Yang, D. He, X. Ge, T. Xu, Y. Wang, H. Qin, and J. Zhang, "Boosting neural representations for videos with a conditional decoder," in *Proceedings of the IEEE/CVF Conference on Computer Vision and Pattern Recognition*, 2024, pp. 2556–2566.

[23] Y. Mao, W. Wang, H. Du, N. Guan, and C. J. Xue, "On the compressibility of quantized large language models," *arXiv preprint arXiv:2403.01384*, 2024.

[24] M. Hershcovitch, L. Choshen, A. Wood, I. Enmouri, P. Chin, S. Sundararaman, and D. Harnik, "Lossless and near-lossless compression for foundation models," *arXiv preprint arXiv:2404.15198*, 2024.

[25] V. Engelson, P. Fritzson, and D. Fritzson, *Lossless compression of high-volume numerical data from simulations*. Linköping University Electronic Press, 2000.

[26] P. Shilane, G. Wallace, M. Huang, and W. Hsu, "Delta compressed and deduplicated storage using stream-informed locality." in *HotStorage*, 2012.

[27] A. Reisizadeh, A. Mokhtari, H. Hassani, A. Jadbabaie, and R. Pedarsani, "Fedpaq: A communication-efficient federated learning method with periodic averaging and quantization," in *International Conference on Artificial Intelligence and Statistics*. PMLR, 2020, pp. 2021–2031.

[28] J. Wang, B. Yuan, L. Rimanic, Y. He, T. Dao, B. Chen, C. Ré, and C. Zhang, "Fine-tuning language models over slow networks using activation quantization with guarantees," *Advances in Neural Information Processing Systems*, vol. 35, pp. 19 215–19 230, 2022.

[29] Z. Hu, X. Zou, W. Xia, S. Jin, D. Tao, Y. Liu, W. Zhang, and Z. Zhang, "Delta-dnn: Efficiently compressing deep neural networks via exploiting floats similarity," in *Proceedings of the 49th International Conference on Parallel Processing*, 2020, pp. 1–12.

[30] Y. Chen, Z. Liu, B. Ren, and X. Jin, "On efficient constructions of checkpoints," *arXiv preprint arXiv:2009.13003*, 2020.

[31] X. Yao and A. Klimovic, "Deltazip: Multi-tenant language model serving via delta compression," *arXiv preprint arXiv:2312.05215*, 2023.

[32] L. Chen, Z. Ye, Y. Wu, D. Zhuo, L. Ceze, and A. Krishnamurthy, "Punica: Multi-tenant lora serving," *Proceedings of Machine Learning and Systems*, vol. 6, pp. 1–13, 2024.

[33] R. Rombach, A. Blattmann, D. Lorenz, P. Esser, and B. Ommer, "High-resolution image synthesis with latent diffusion models," in *Proceedings of the IEEE/CVF Conference on Computer Vision and Pattern Recognition (CVPR)*, June 2022, pp. 10 684–10 695.

[34] A. Radford, J. Wu, R. Child, D. Luan, D. Amodei, and I. Sutskever, "Language models are unsupervised multitask learners," 2019.

[35] J. Devlin, M. Chang, K. Lee, and K. Toutanova, "BERT: pre-training of deep bidirectional transformers for language understanding," *CoRR*, vol. abs/1810.04805, 2018. [Online]. Available: http://arxiv.org/abs/1810.04805

[36] K. He, X. Zhang, S. Ren, and J. Sun, "Deep residual learning for image recognition," in *Proceedings of the IEEE conference on computer vision and pattern recognition*, 2016, pp. 770–778.

[37] Lambda, "lambdalabs/sd-pokemon-diffusers." https://huggingface.co/lambdalabs/sd-pokemon-diffusers, huggingFace model.

[38] N. . LTI/CMU, "neulab/gpt2-finetuned-wikitext103." https://huggingface.co/neulab/gpt2-finetuned-wikitext103, huggingFace model.

[39] AssemblyAI. assemblyai/bert-large-uncased-sst2. [Online]. Available: https://huggingface.co/assemblyai/bert-large-uncased-sst2

[40] C. xcx. Celal11/resnet-50-finetuned-fer2013-0.0033. [Online]. Available: https://huggingface.co/Celal11/resnet-50-finetuned-FER2013-0.003

[41] X. Li, K. Huang, W. Yang, S. Wang, and Z. Zhang, "On the convergence of fedavg on non-iid data," *arXiv preprint arXiv:1907.02189*, 2019.

[42] C. T. Dinh, N. H. Tran, M. N. Nguyen, C. S. Hong, W. Bao, A. Y. Zomaya, and V. Gramoli, "Federated learning over wireless networks: Convergence analysis and resource allocation," *IEEE/ACM Transactions on Networking*, vol. 29, no. 1, pp. 398–409, 2020.

[43] L. Bottou, "Stochastic gradient descent tricks," *Neural Networks: Tricks of the Trade: Second Edition*, pp. 421–436, 2012.

[44] S. Ben-David, J. Blitzer, K. Crammer, and F. Pereira, "Analysis of representations for domain adaptation," *Advances in neural information processing systems*, vol. 19, 2006.

[45] L. Chen, F. Yuan, J. Yang, X. He, C. Li, and M. Yang, "User-specific adaptive fine-tuning for cross-domain recommendations," *IEEE Transactions on Knowledge and Data Engineering*, vol. 35, no. 3, pp. 3239–3252, 2021.

[46] Y. Cui, Y. Song, C. Sun, A. Howard, and S. Belongie, "Large scale fine-grained categorization and domain-specific transfer learning," in *Proceedings of the IEEE conference on computer vision and pattern recognition*, 2018, pp. 4109–4118.

[47] J. H. Wilkinson, "Error analysis of floating-point computation," *Numerische Mathematik*, vol. 2, pp. 319–340, 1960.

[48] M. Schindler, "Range encoder version 1.3, 2000." http://www.compressconsult.com/rangecoder/, october 1999.

[49] W. Kahan, "Ieee standard 754 for binary floating-point arithmetic," *Lecture Notes on the Status of IEEE*, vol. 754, no. 94720-1776, p. 11, 1996.

[50] A. Paszke, S. Gross, F. Massa, A. Lerer, J. Bradbury, G. Chanan, T. Killeen, Z. Lin, N. Gimelshein, L. Antiga *et al.*, "Pytorch: An imperative style, high-performance deep learning library," *Advances in neural information processing systems*, vol. 32, 2019.

[51] E. Almazrouei, H. Alobeidli, A. Alshamsi, A. Cappelli, R. Cojocaru, M. Debbah, E. Goffinet, D. Heslow, J. Launay, Q. Malartic, B. Noune, B. Pannier, and G. Penedo, "Falcon-40B: an open large language model with state-of-the-art performance," 2023.

[52] S. Black, S. Biderman, E. Hallahan, Q. Anthony, L. Gao, L. Golding, H. He, C. Leahy, K. McDonell, J. Phang, M. Pieler, U. S. Prashanth, S. Purohit, L. Reynolds, J. Tow, B. Wang, and S. Weinbach, "Gpt-neox-20b: An open-source autoregressive language model," 2022. [Online]. Available: https://arxiv.org/abs/2204.06745

[53] B. Wang and A. Komatsuzaki, "GPT-J-6B: A 6 Billion Parameter Autoregressive Language Model," https://github.com/kingoflolz/mesh-transformer-jax, May 2021.

[54] K. Lagler, M. Schindelegger, J. Böhm, H. Krásná, and T. Nilsson, "Gpt2: Empirical slant delay model for radio space geodetic techniques," *Geophysical research letters*, vol. 40, no. 6, pp. 1069–1073, 2013.

[55] M. P. Marcus, B. Santorini, and M. A. Marcinkiewicz, "Building a large annotated corpus of English: The Penn Treebank," *Computational Linguistics*, vol. 19, no. 2, pp. 313–330, 1993. [Online]. Available: https://www.aclweb.org/anthology/J93-2004

[56] S. Merity, C. Xiong, J. Bradbury, and R. Socher, "Pointer sentinel mixture models," 2016.

[57] C. Chelba, T. Mikolov, M. Schuster, Q. Ge, T. Brants, P. Koehn, and T. Robinson, "One billion word benchmark for measuring progress in statistical language modeling," 2014.

[58] J. L. Grisman, "Jorgeutd/bert-large-uncased-finetuned-ner." https://huggingface.co/Jorgeutd/bert-large-uncased-finetuned-ner, huggingFace model.

[59] P. N. group. princeton-nlp/sup-simcse-bert-large-uncased. [Online]. Available: https://huggingface.co/princeton-nlp/sup-simcse-bert-large-uncased

[60] ——. princeton-nlp/unsup-simcse-bert-large-uncased. [Online]. Available: https://huggingface.co/princeton-nlp/unsup-simcse-bert-large-uncased

[61] Y. Matsubara. yoshitomo-matsubara/bert-large-uncased-mrpc. [Online]. Available: https://huggingface.co/yoshitomo-matsubara/bert-large-uncased-mrpc

[62] E. F. Coutinho, F. R. de Carvalho Sousa, P. A. L. Rego, D. G. Gomes, and J. N. de Souza, "Elasticity in cloud computing: a survey," *annals of telecommunications-annales des télécommunications*, vol. 70, pp. 289–309, 2015.

[63] A. Shahbahrami, R. Bahrampour, M. S. Rostami, and M. A. Mobarhan, "Evaluation of huffman and arithmetic algorithms for multimedia compression standards," *arXiv preprint arXiv:1109.0216*, 2011.

[64] A. Wang, A. Singh, J. Michael, F. Hill, O. Levy, and S. R. Bowman, "GLUE: A multi-task benchmark and analysis platform for natural language understanding," 2019, in the Proceedings of ICLR.

[65] M. A. Lab. (2020, January) Bean disease dataset. [Online]. Available: https://github.com/AI-Lab-Makerere/ibean/

[66] Hugging Face, "bert-base-cased," https://huggingface.co/bert-base-cased, 2018.

[67] ——, "google/vit-base-patch16-224-in21k," https://huggingface.co/google/vit-base-patch16-224-in21k, 2020.

[68] ——, "Transformers: State-of-the-art natural language processing for PyTorch and TensorFlow," https://github.com/huggingface/transformers, 2022.

[69] KoboldAI, "Koboldai/gpt-neox-20b-erebus." https://huggingface.co/KoboldAI/GPT-NeoX-20B-Erebus, huggingFace model.

# A Extended Statistics from HuggingFace

In Table 8, we provide the statistical results of the ten more model families and show the proportion of full models on these families, further underscoring that the number of full fine-tuned models still occupy the majority in the HuggingFace repository. In Table 9, we roughly calculate the portion of pre-trained and fine-tuned models in HuggingFace. Specifically, based on model creation time, we iterate 10,000 models respectively in both ascending (old to new) and descending (new to old) order. Among these models, we only count the models that have explicitly stated their identity (i.e., pre-trained or fine-tuned) in their "README.md" file. We can see that fine-tuned models occupy a significant portion (81% and 99%) of the model hub. Furthermore, the results of the descending order indicates that fine-tuned models have become overwhelmingly dominant currently.

Table 8: The number of full fine-tuned and PEFT models in the ten additional model families, along with the proportion of full models on these families.

| Model | # Full | # PEFT | Proportion of Full |
|---|---|---|---|
| Gemma-9b | 315 | 121 | 72% |
| Gemma-2b | 3,836 | 279 | 93% |
| Bloom-7b1 | 163 | 105 | 60% |
| Bloom-1b7 | 130 | 61 | 68% |
| Pythia-12b | 120 | 131 | 47% |
| Pythia-6.9b | 316 | 93 | 77% |
| T5-xxl | 106 | 62 | 63% |
| T5-large | 1,277 | 203 | 86% |
| Llama-2-70b | 214 | 96 | 69% |
| Mistral-7b | 6,972 | 2,027 | 77% |
| **AVG** | | | **71%** |

Table 9: The portion of pre-trained and fine-tuned models in the 10,000 models from HuggingFace, counted in ascending and descending order.

| | Ascending (old to new) | | Descending (new to old) | |
|---|---|---|---|---|
| | **# Pretrained** | **# Finetuned** | **# Pretrained** | **# Finetuned** |
| **Num.** | 501 | 2,082 | 52 | 4,295 |
| **Portion** | 19% | 81% | 1% | 99% |

# B Detailed Empirical Results about Delta

Here we present the detailed empirical results about delta between fine-tuned and pre-trained models. Figure 10 shows the detailed residual matrix of different layers on Wikitext103 GPT-2. Figure 9 shows that the average parameter element difference on four model families, which grows with a slow speed as the fine-tuning processes.

# C Theoretical Derivations

## C.1 Proof of Theorem 1

Before deriving Theorem 1, we first relate the gradient and the loss through the following derivation.

Since $f$ is $\beta$-smooth, for any $\mathbf{w}, \mathbf{v} \in \mathbb{R}^d$, we have

$$f(\mathbf{w}) - f(\mathbf{v}) - \nabla f(\mathbf{v})^\top (\mathbf{w} - \mathbf{v}) \leq \frac{\beta}{2} \|\mathbf{w} - \mathbf{v}\|^2.$$

For one step of gradient descent $\mathbf{w}_{t+1} = \mathbf{w}_t - \frac{1}{\beta} G(\mathbf{w}_t)$, we have

$$f(\mathbf{w}_{t+1}) - f(\mathbf{w}_t) \leq \nabla f(\mathbf{w}_t)^\top (\mathbf{w}_{t+1} - \mathbf{w}_t) + \frac{\beta}{2} \|\mathbf{w}_{t+1} - \mathbf{w}_t\|^2$$

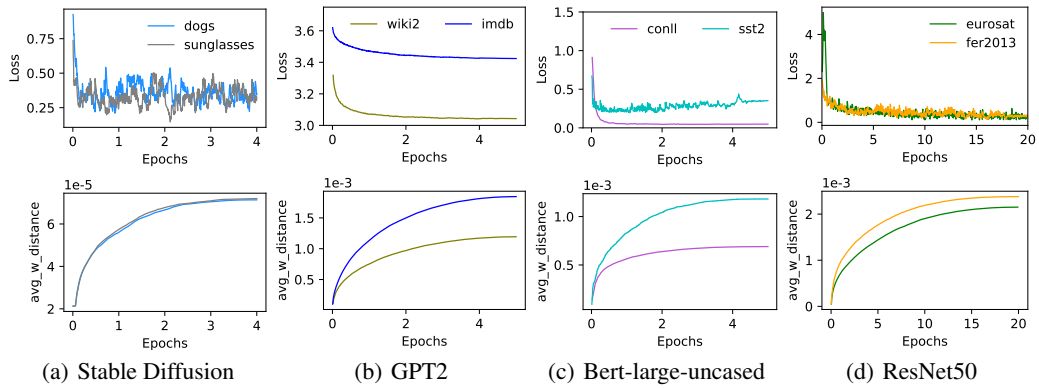

Figure 9: Fine-tuning results on different models.

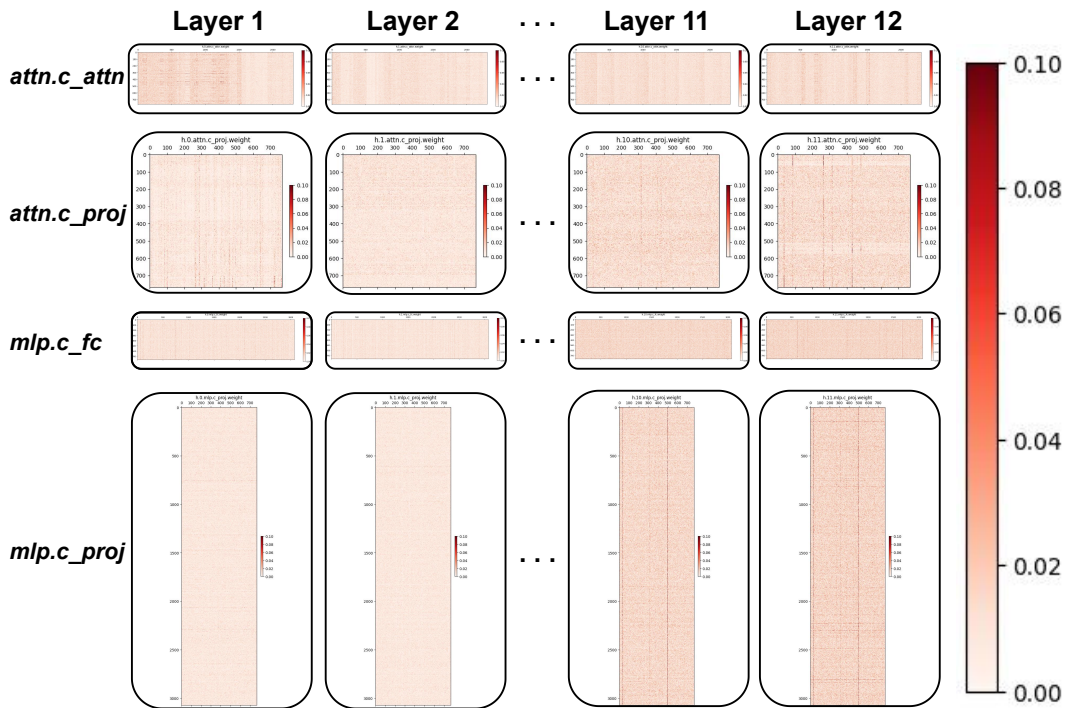

Figure 10: Residual matrix of GPT-2 on Wikitext103.

$$\leq \nabla f(\mathbf{w}_t)^T (-\eta G(\mathbf{w}_t)) + \frac{\beta\eta^2}{2}||G(\mathbf{w}_t)||^2.$$

Taking the expectation before $t$, we have

$$\mathbb{E}\left[f(\mathbf{w}_{t+1})\right] - \mathbb{E}\left[f(\mathbf{w}_t)\right]$$

$$\leq \nabla f(\mathbf{w}_t)^T (-\eta \nabla f(\mathbf{w}_t)) + \frac{\beta\eta^2}{2}\mathbb{E}\left[||G(\mathbf{w}_t)||^2\right]$$

$$\leq -\eta||\nabla f(\mathbf{w}_t)||^2 + \frac{\beta\eta^2}{2}\left(||\mathbb{E}\left[G(\mathbf{w}_t)\right]||^2 + \sigma^2\right)$$

$$\leq (\frac{\beta\eta^2}{2} - \eta)||\nabla f(\mathbf{w}_t)||^2 + \frac{\beta\eta^2\sigma^2}{2}. \tag{7}$$

Let $g(\mathbf{w}) = f(\mathbf{w}) - f(\mathbf{w}^*) \geq 0$. The gradient of $g(\mathbf{w})$ is $\nabla g(\mathbf{w}) = \nabla f(\mathbf{w}) - \nabla f(\mathbf{w}^*) = \nabla f(\mathbf{w})$. Therefore, $g(\mathbf{w})$ also satisfies the assumption of bounded variance $\mathbb{E}_{\xi\sim\mathcal{D}}\left\|G(\mathbf{w};\xi) - \nabla f(\mathbf{w})\right\|^2 \leq \sigma^2$. Substituting $g(\mathbf{w})$ into the above inequality, we have

$$\mathbb{E}\left[g(\mathbf{w}_{t+1}) - g(\mathbf{w}_t)\right] \leq (\frac{\beta\eta^2}{2} - \eta)||\nabla g(\mathbf{w}_t)||^2 + \frac{\beta\eta^2\sigma^2}{2}$$

$$-\mathbb{E}\left[g(\mathbf{w})\right] \leq (\frac{\beta\eta^2}{2} - \eta)||\nabla g(\mathbf{w}_t)||^2 + \frac{\beta\eta^2\sigma^2}{2}.$$

We arrange the above inequality and get

$$\left(2\eta - \beta\eta^2\right)||\nabla f(\mathbf{w}) - \nabla f(\mathbf{w}^*)||^2 \leq 2\left(f(\mathbf{w}) - f(\mathbf{w}^*) - \nabla f(\mathbf{w}^*)^T(\mathbf{w} - \mathbf{w}^*)\right) + \beta\eta^2\sigma^2.$$

Since $0 < \eta < \frac{2}{\beta}$, we have $2\eta - \beta\eta^2 > 0$. Therefore, the above inequality can be further written as

$$||\nabla f(\mathbf{w})||^2 \leq \frac{2}{2\eta - \beta\eta^2}\left(f(\mathbf{w}) - f(\mathbf{w}^*)\right) + \frac{\beta\eta\sigma^2}{2 - \beta\eta}.$$

Thus, we have related the gradient $\nabla f(\mathbf{w})$ to the loss $f(\mathbf{w})$. Then give the proof of Theorem 1 as the following:

*Proof.* Since $\mathbf{w}_f = \mathbf{w}_p - \sum_{t=0}^{T}\eta_t G(\mathbf{w}_t)$, we have

$$||\mathbf{w}_f - \mathbf{w}_p||^2$$

$$\leq \sum_{t=1}^{T}||\eta_t\left(G(\mathbf{w}_t) - \nabla f(\mathbf{w}_t) + \nabla f(\mathbf{w}_t)\right)||^2$$

$$\leq \sum_{t=1}^{T}\eta_t^2\left(\sigma^2 + ||\nabla f(\mathbf{w}_t)||^2\right)$$

$$\overset{(a)}{\leq} \sum_{t=1}^{T}\eta_t^2\sigma^2 + \sum_{t=1}^{T}\eta_t^2\left(\frac{2}{2\eta_t - \beta\eta_t^2}\left(f(\mathbf{w}) - f(\mathbf{w}^*)\right) + \frac{\beta\eta_t\sigma^2}{2 - \beta\eta_t}\right)$$

$$= \underbrace{\sum_{t=1}^{T}\frac{\sigma^2}{\beta^2 t} + \sum_{t=1}^{T}\frac{\sigma^2}{2\beta^2}\cdot\frac{1}{t(\sqrt{t} - \frac{1}{2})}}_{(I)} + \underbrace{\sum_{t=1}^{T}\left(\frac{1}{\beta\left(\sqrt{t} - \frac{1}{2}\right)}\left(f(\mathbf{w}) - f(\mathbf{w}^*)\right)\right)}_{(II)}, \tag{8}$$

where (a) follows from the above remark.

For the item $(I)$, we have

$$\sum_{t=1}^{T} \frac{\sigma^2}{\beta^2 t} + \sum_{t=1}^{T} \frac{\sigma^2}{2\beta^2} \cdot \frac{1}{t(\sqrt{t} - \frac{1}{2})}$$

$$= \sum_{t=1}^{T} \frac{\sigma^2}{\beta^2} \left( \frac{1}{t} + 2 \cdot \left( \frac{1}{\sqrt{t} - \frac{1}{2}} - \frac{1}{\sqrt{t}} - \frac{\frac{1}{2}}{t} \right) \right)$$

$$\leq \frac{2\sigma^2}{\beta^2} + \sum_{t=2}^{T} \frac{2\sigma^2}{\beta^2} \left( \frac{1}{\sqrt{t} - \frac{1}{2}} - \frac{1}{\sqrt{t} + \frac{1}{2}} \right)$$

$$= \frac{2\sigma^2}{\beta^2} + \sum_{t=2}^{T} \frac{2\sigma^2}{\beta^2} \cdot \frac{1}{t - \frac{1}{4}}$$

$$\overset{(a)}{\leq} \frac{2\sigma^2}{\beta^2} + \frac{2\sigma^2}{\beta^2} \cdot \int_{1}^{T} \frac{1}{t - \frac{1}{4}} dt$$

$$\leq \frac{2\sigma^2}{\beta^2} + \frac{2\sigma^2}{\beta^2} \cdot \left( \ln\left(T - \frac{1}{4}\right) - \ln(\frac{3}{4}) \right),$$

where (a) follows from that for a monotonically decreasing function $f$ over the region $[a - 1, b + 1]$, it holds that $\int_{a-1}^{b} f(x)\mathrm{d}x > \sum_{i=a}^{b} f(i)$.

For the item $(II)$, we have

$$\sum_{t=1}^{T} \left( \frac{1}{\beta \left( \sqrt{t} - \frac{1}{2} \right)} (f(\mathbf{w}_t) - f(\mathbf{w}^*)) \right)$$

$$= \sum_{t=1}^{T} \frac{1}{\beta \left( \sqrt{t} - \frac{1}{2} \right)} (f(\mathbf{w}_t) - f(\mathbf{w}_p)) + \frac{2}{\beta}(f(\mathbf{w}_p) - f(\mathbf{w}^*)) + \sum_{t=2}^{T} \frac{1}{\beta \left( \sqrt{t} - \frac{1}{2} \right)} (f(\mathbf{w}_p) - f(\mathbf{w}^*))$$

$$\overset{(a)}{\leq} \sum_{t=1}^{T} \frac{1}{\beta \left( \sqrt{t} - \frac{1}{2} \right)} (f(\mathbf{w}_t) - f(\mathbf{w}_p)) + \frac{2}{\beta}(f(\mathbf{w}_p) - f(\mathbf{w}^*)) + \frac{f(\mathbf{w}_p) - f(\mathbf{w}^*)}{\beta} \int_{1}^{T} \frac{1}{\sqrt{t} - \frac{1}{2}} dt$$

$$= \sum_{t=1}^{T} \frac{1}{\beta \left( \sqrt{t} - \frac{1}{2} \right)} (f(\mathbf{w}_t) - f(\mathbf{w}_p)) + \frac{-2 + \ln 2 + 2\sqrt{T} + \ln(\sqrt{T} - \frac{1}{2})}{\beta}(f(\mathbf{w}_p) - f(\mathbf{w}^*)),$$

where (a) follows from that for a monotonically decreasing function $f$ over the region $[a - 1, b + 1]$, it holds that $\int_{a-1}^{b} f(x)\mathrm{d}x > \sum_{i=a}^{b} f(i)$.

Taking the expectation and summing up the inequality (7) for $t$ steps, we have

$$\sum_{t=1}^{T} \left( \frac{1}{\beta \left( \sqrt{t} - \frac{1}{2} \right)} \mathbb{E}\left(f(\mathbf{w}_t) - f(\mathbf{w}^*)\right) \right)$$

$$\leq \sum_{t=1}^{T} \frac{1}{\beta \left( \sqrt{t} - \frac{1}{2} \right)} \cdot \frac{\sigma^2}{2\beta} + \frac{-2 + \ln 2 + 2\sqrt{T} + \ln(\sqrt{T} - \frac{1}{2})}{\beta}(f(\mathbf{w}_p) - f(\mathbf{w}^*))$$

$$\leq \frac{\sigma^2}{\beta^2} + \frac{\sigma^2}{2\beta^2} \cdot \int_{1}^{T} \frac{1}{\sqrt{t} - \frac{1}{2}} dt + \frac{-2 + \ln 2 + 2\sqrt{T} + \ln(\sqrt{T} - \frac{1}{2})}{\beta}(f(\mathbf{w}_p) - f(\mathbf{w}^*))$$

$$= \frac{\sigma^2}{2\beta^2} \cdot \left( \ln 2 + 2\sqrt{T} + \ln(\sqrt{T} - \frac{1}{2}) \right) + \frac{-2 + \ln 2 + 2\sqrt{T} + \ln(\sqrt{T} - \frac{1}{2})}{\beta}(f(\mathbf{w}_p) - f(\mathbf{w}^*))$$

Combining the above two items, we can write the expectation of the inequality (8) as

$$\mathbb{E}\left[||\mathbf{w}_f - \mathbf{w}_p||^2\right]$$

$$\leq \frac{2\sigma^2}{\beta^2}\left(1+2\ln 2 - \ln 3 + \ln\left(T - \frac{1}{4}\right)\right) + \frac{\sigma^2}{2\beta^2}\cdot\left(\ln 2 + 2\sqrt{T} + \ln(\sqrt{T} - \frac{1}{2})\right)$$

$$+ \frac{-2 + \ln 2 + 2\sqrt{T} + \ln(\sqrt{T} - \frac{1}{2})}{\beta}(f(\mathbf{w}_p) - f(\mathbf{w}^*))$$

$$= \frac{\sigma^2}{2\beta^2}(4 + 9\ln 2 - 4\ln 3) + \frac{\sigma^2}{2\beta^2}\left(4\ln(T - \frac{1}{4}) + 2\sqrt{T} + \ln(\sqrt{T} - \frac{1}{2})\right) \qquad (9)$$

$$+ \frac{-2 + \ln 2 + 2\sqrt{T} + \ln(\sqrt{T} - \frac{1}{2})}{\beta}(f(\mathbf{w}_p) - f(\mathbf{w}^*))$$

$$\leq \frac{3\sigma^2}{\beta^2} + \frac{\sigma^2}{2\beta^2}\left(4\ln(T) + 2\sqrt{T} + \frac{1}{2}\ln(T)\right) + \frac{2\sqrt{T} + \frac{1}{2}\ln(T)}{\beta}(f(\mathbf{w}_p) - f(\mathbf{w}^*)).$$

From the expression of variance $D(\mathbf{x}) = \mathbb{E}\left[||\mathbf{x}||^2\right] - \mathbb{E}\left[||\mathbf{x}||\right]^2$, we have

$$\mathbb{E}\left[||\mathbf{w}_f - \mathbf{w}_p||\right] \leq \frac{\sqrt{3}\sigma}{\beta} + \sqrt{\frac{9\sigma^2}{4\beta^2} + \frac{f(\mathbf{w}_p) - f(\mathbf{w}^*)}{2\beta}}\cdot\sqrt{\ln(T)} + \sqrt{\frac{\sigma^2}{\beta^2} + \frac{2(f(\mathbf{w}_p) - f(\mathbf{w}^*))}{\beta}}\cdot T^{\frac{1}{4}}$$

This completes the proof of Theorem 1. $\qquad\square$

## C.2 Theorem 1 in Domain adaptation

We first give some definitions and assumptions as the previous works(44) in domain adaptation. Let $\mathcal{X} \subset \mathbb{R}^p$ be the input space and $\mathcal{Y} \subset \mathbb{R}$ be the output space. $\mathcal{T}$ denotes a domain (or task), which consists of a data distribution $\mathcal{D}$ over $\mathcal{X}$. We consider a binary classification task with the hypothesis class and the loss function is Huber loss $l(\hat{y}, y) = |\hat{y} - y|$. Let $h : \mathcal{X} \to \mathcal{Y}$ denote a hypothesis that maps inputs to predicted labels, and $\mathcal{H} \subseteq \{h : \mathcal{X} \to \mathcal{Y}\}$ denote a hypothesis class. The $\mathcal{H} \triangle \mathcal{H} := \{h(x) \oplus h'(x), h, h' \in \mathcal{H}\}$ is defined as the symmetric difference hypothesis space, where $\oplus$ denotes the XOR operator. $\mathcal{A}_{\mathcal{H}\triangle\mathcal{H}}$ is a set of measurable subsets for $\forall h(x) \oplus h'(x) \in \mathcal{H} \triangle \mathcal{H}$. Then $d_{\mathcal{H}\triangle\mathcal{H}}(\mathcal{D}, \mathcal{D}') := 2\sup_{\mathcal{A}\in\mathcal{A}_{\mathcal{H}\triangle\mathcal{H}}}|Pr_{\mathcal{D}}(\mathcal{A}) - Pr_{\mathcal{D}'}(\mathcal{A})|$ is defined as the distribution divergence induced by the symmetric difference hypothesis space, given two distributions $\mathcal{D}$ and $\mathcal{D}'$.

From the generation bound in (44), we first give a variant based on the two data distributions of the source and target domain. Let $\mathcal{T}_S$ and $\mathcal{T}_T$ be the source and target domains, whose data distributions are $\mathcal{D}_S$ and $\mathcal{D}_T$ respectively. Let $\mathcal{H} \subseteq \{h : \mathcal{X} \to \mathcal{Y}\}$ denote a hypothesis class with VC-dimension $d$. Then with probability at least $1 - \delta, \forall h \in \mathcal{H}$:

$$\epsilon_T(h) \leq \hat{\epsilon}_S(h) + \sqrt{\frac{4}{m}\left(d\log\frac{2em}{d} + \log\frac{4}{\delta}\right)} + d_{\mathcal{H}\triangle\mathcal{H}}(\mathcal{D}_S, \mathcal{D}_T) + \lambda, \qquad (10)$$

where $e$ is the base of the natural logarithm, $\hat{\epsilon}_S(h) = \mathbb{E}_{xy\sim\mathcal{D}_S}|h(x) - y|$ is the empirical error of the source domain based on $m$ observable samples, and $\lambda = \min_{h\in\mathcal{H}}(\epsilon_T(h) + \epsilon_S(h))$ is the optimal error on the two domains.

We regard the pre-trained and fine-tuned domains as the source and target domains respectively. The pre-trained model with parameters $\mathbf{w}_p$ can be regarded as the hypothesis. Therefore, based on Inequality (10), we have

$$f_{\mathcal{T}_F}(\mathbf{w}_p) \leq \hat{f}_{\mathcal{T}_P}(\mathbf{w}_p) + \sqrt{\frac{4}{m}\left(d\log\frac{2em}{d} + \log\frac{4}{\delta}\right)} + d_{\mathcal{H}\triangle\mathcal{H}}(\mathcal{D}_P, \mathcal{D}_F) + \lambda. \qquad (11)$$

Inequality (9) can be equivalently written as

$$\mathbb{E}\left[||\mathbf{w}_f - \mathbf{w}_p||^2\right] \leq \frac{3\sigma^2}{\beta^2} + \frac{\sigma^2}{2\beta^2}\left(\frac{9}{2}\ln T + 2\sqrt{T}\right) + \frac{f_{\mathcal{T}_F}(\mathbf{w}_p) - f_{\mathcal{T}_F}(\mathbf{w}^*)}{\beta}$$

$$\cdot\left(\frac{1}{2}\ln T + 2\sqrt{T}\right).$$

Applying (11) to the above inequality, we relate the model difference to the data distribution divergence between the fine-tuned and pre-trained domains.

## C.3 Proof of Theorem 2

*Proof.* We first give the most widely used IEEE floating-point standard representation:

$$n = (-1)^s \times m \times 2^e,$$

where $s$ is the sign, $e$ is the exponent value using a biased representation, and $m$ is the fractional part. If we map $w_f$ and $w_p$ into the integers $\hat{w}_f$ and $\hat{w}_p$ while keeping the same bit stream, we have

$$
\begin{aligned}
r &= \left\lceil \log_2 \hat{\Delta} \right\rceil \\
&= \left\lceil \log_2 \left( 2^{(n_e + n_m)} (s_f \oplus s_p) + 2^{n_m} (e_f - e_p) + \hat{m}_f - \hat{m}_p \right) \right\rceil.
\end{aligned}
$$

When $w_f$ and $w_p$ have the same sign (i.e., $s_f \oplus s_p = 0$), we have

$$
\begin{aligned}
r &= \left\lceil \log_2 \left( 2^{n_m} (e_f - e_p) + \hat{m}_f - \hat{m}_p \right) \right\rceil \\
&\leq \left\lceil \log_2 \left( 2^{n_m} (e_f - e_p) + 2^{n_m} \right) \right\rceil \\
&= \left\lceil n_m + \log_2 (e_f - e_p + 1) \right\rceil \\
&\stackrel{(a)}{=} \left\lceil n_m + \log_2 \left( \log_2 \left( \frac{w_f}{w_p} \cdot \frac{m_p}{m_f} \right) + 1 \right) \right\rceil \\
&\stackrel{(b)}{\leq} \left\lceil n_m + \log_2 \left( \log_2 \left( \frac{w_f}{w_p} \cdot 2 \right) + 1 \right) \right\rceil \\
&= \left\lceil n_m + \log_2 \left( \log_2 \left( \frac{w_f}{w_p} \right) + 2 \right) \right\rceil,
\end{aligned}
$$

where (a) is from that $\frac{w_f}{w_p} = \frac{2^{e_f} \cdot m_f}{2^{e_p} \cdot m_p} = 2^{(e_f - e_p)} \cdot \frac{m_f}{m_p}$; (b) is from that the fraction $m$ of float is in $[1, 2)$. This completes the proof of Theorem 2. $\qquad\square$

## D   Algorithm Details

Table 10 provides the mapping examples of `FM-Delta`. We detail the mapping process as the following two steps. Firstly, convert floats to signed integers while retaining the same byte string. Secondly, monotonically map signed integers into unsigned integers. For positive values, the representation range is $[00000000..., 01111111...)$, i.e. $[0, 2^{31})$. Both signed and unsigned integers are monotonically increasing in this interval. We just need to map the $[0, 2^{31})$ of signed integers to the $[2^{31}, 2^{32})$ of unsigned integers. Therefore, we flip the most significant bit for positive values, so the range becomes $[10000000..., 11111111...)$. For negative values, the representation range is $[11111111..., 10000000...)$, i.e. $[-2^{31}, 0)$. The monotonicity of signed and unsigned integers is opposite. Therefore, we flip all the bits of the negative integer to meet the alignment of monotonicity, so the range becomes $[00000000..., 01111111...)$.

Table 10: Example of mapping positive and negative floats.

| Float | Signed Integer | Unsigned Integer |
|---|---|---|
| 0.0316 | 00111101000000010110111100000000 | 10111101000000010110111100000000 |
| -0.0316 | 10111101000000010110111100000000 | 01000010111111101001000011111111 |

## E   Experiment Details

### E.1   Model Compression Details

Table 11 presents the detailed information of the top-5 fine-tuned models in the seven families from HuggingFace, corresponding to the compression results in Figure 6.

### E.2   Compression Rates on Different Layers

We present the compression rates of `FM-Delta` on different model layers in Figure 11, which shows that for a specific model, the compression rate on each layer generally fluctuates around a certain value.

Table 11: Model details of the top-5 fine-tuned models in the seven families from HuggingFace, and the single compression rates of `FM-Delta` on them.

| Family | Model Type | Datasets | Model ID in HuggingFace | Comp. Rate |
|---|---|---|---|---|
| Falcon-40B | Pretrained | RefinedWeb | tiiuae/falcon-40b | |
| | Finetuned | Baize | tiiuae/falcon-40b-instruct | 50% |
| | | OAS.& Dol.&Syt. | OpenAssistant/falcon-40b-sft-mix-1226 | 53% |
| | | OASST | h2oai/h2ogpt-oasst1-falcon-40b | 43% |
| | | WizardLM | ehartford/WizardLM-Uncensored-Falcon-40b | 48% |
| | | Alpaca&Dolly | falcon-40b-ft-alpaca-dolly-dutch | 48% |
| GPT-NeoX-20B | Pretrained | The Pile | EleutherAI/gpt-neox-20b | |
| | Finetuned | Erebus | KoboldAI/GPT-NeoX-20B-Erebus | 30% |
| | | Chat | togethercomputer/GPT-NeoXT-Chat-Base-20B | 42% |
| | | Skein | KoboldAI/GPT-NeoX-20B-Skein | 28% |
| | | OASST | dvruette/oasst-gpt-neox-20b-1000-steps | 46% |
| | | Instruction | jordiclive/instruction-tuned-gpt-neox-20b | 53% |
| GPT-J-6B | Pretrained | The Pile | EleutherAI/gpt-j-6b | |
| | Finetuned | Janeway | KoboldAI/GPT-J-6B-Janeway | 52% |
| | | CoT&P3&NI | togethercomputer/GPT-JT-6B-v1 | 60% |
| | | OASST | reciprocate/gpt-j_rm_format-oa | 55% |
| | | Skein | KoboldAI/GPT-J-6B-Skein | 56% |
| | | Adventure | KoboldAI/GPT-J-6B-Adventure | 59% |
| GPT-2 | Pretrained | WebText | gpt2 | |
| | Finetuned | IMDB | rajkumarrrk/gpt2-fine-tuned-on-imdb-positive-reviews | 68% |
| | | Wikitext103 | neulab/gpt2-finetuned-wikitext103 | 75% |
| | | CNN-DailyMail | gavin124/gpt2-finetuned-cnn-summarization-v2 | 78% |
| | | CommonGen | mrm8488/GPT-2-finetuned-common_gen | 71% |
| | | SQuAD | anas-awadalla/gpt2-span-head-finetuned-squad | 62% |
| Bert-large-uncased | Pretrained | BookCorpus | bert-large-uncased | |
| | Finetuned | CoNLL2003 | Jorgeutd/bert-large-uncased-finetuned-ner | 68% |
| | | MNLI&SNLI | princeton-nlp/sup-simcse-bert-large-uncased | 63% |
| | | Wikipedia | princeton-nlp/unsup-simcse-bert-large-uncased | 65% |
| | | SST2 | assemblyai/bert-large-uncased-sst2 | 63% |
| | | MRPC | yoshitomo-matsubara/bert-large-uncased-mrpc | 58% |
| Stable Diffusion | Pretrained | LAION-Aesthetics | runwayml/stable-diffusion-v1-5 | |
| | Finetuned | Waltz | mikesmodels/Waltz_with_Bashir_Diffusion | 59% |
| | | Oscar | iriscope/oscarvatar | 62% |
| | | CloneWars | questcoast/clone-wars-diffusion-v1 | 63% |
| | | SJH | ProGamerGov/Min-Illust-Background-Diffusion | 60% |
| | | Kurzgesagt | questcoast/SD-Kurzgesagt-style-finetune | 58% |
| ResNet50 | Pretrained | ImageNet-1k | microsoft/resnet-50 | |
| | Finetuned | Memes | jayanta/resnet50-finetuned-memes | 66% |
| | | BrainTumor | Alia-Mohammed/resnet-50-finetuned-brain-tumor | 67% |
| | | FER2013 | Celal11/resnet-50-finetuned-FER2013-0.003 | 83% |
| | | Eurosat | keithanpai/resnet-50-finetuned-eurosat | 59% |
| | | NCT-CRC-HE-100K | polejowska/resnet-50-finetuned-nct-crc-he-45k | 69% |

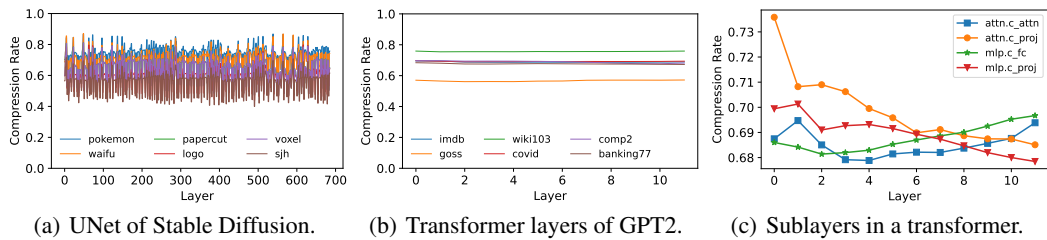

(a) UNet of Stable Diffusion.  (b) Transformer layers of GPT2.  (c) Sublayers in a transformer.

Figure 11: Compression rates of `FM-Delta` on different model layers.

### E.3 Further Discussion on Throughput

Table 4 shows that `FM-Delta` ranks third in decompression throughput, lower than Zlib and Gzip. Here we give a further explanation about the reason.

Among all the compressors, Zlib and Gzip have similar performances since both of them use the DEFLATE algorithm, which consists of two main components: LZ77 and Huffman coding. Differently, `FM-Delta` is mainly based on the compression algorithm specifically for floating-point data, which uses the range coding to compress array data. Since range coding involves dealing with floating-point numbers and maintaining the current encoding interval, Huffman coding typically has a faster decompression speed than range coding(63), resulting in a higher decompression throughput of Zlib and Gzip.

### E.4 Extended Fine-tuning Results

In Figure 12, we present the additional fine-tuning results on GPT2-1.5B(34). Since the model is bigger than that in the main paper, it can be seen that it overfits on the relatively simple datasets, including PTB(55) and Wikitext2(56). The simplicity of the two datasets results in their larger model difference due to their larger distribution divergence with the original data for training GPT2-1.5B, which is also analyzed in Section 3.

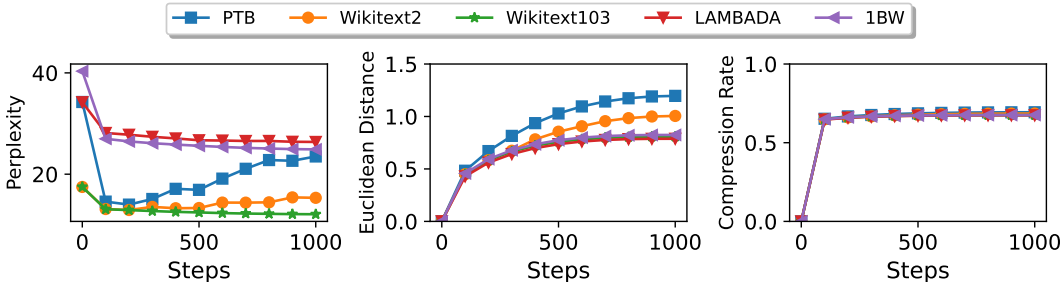

Figure 12: Three metrics over the iteration steps $T$ when fine-tuning GPT2-1.5B on different datasets.

Table 12: Finetuning results for *Bert-base-cased* on MRPC and SST2, and *google/vit-base-patch16-224-in21k* on Beans.

| Task | Steps | Loss | Euc. Distance | Comp. Rate |
|---|---|---|---|---|
| Bert-base-cased on MRPC(64) | 0 | 0.95 | 0 | - |
| | 400 | 0.38 | 0.66 | 46.9% |
| | 800 | 0.21 | 0.71 | 48.5% |
| | 1200 | 0.14 | 0.74 | 48.9% |
| Bert-base-cased on SST2(64) | 0 | 0.74 | 0 | - |
| | 400 | 0.34 | 0.65 | 45.3% |
| | 800 | 0.28 | 0.69 | 46.5% |
| | 1200 | 0.25 | 0.71 | 47.2% |
| google/vit-base-patch16-224-in21k on Beans(65) | 0 | 1.03 | 0 | - |
| | 400 | 0.15 | 0.21 | 62.5% |
| | 800 | 0.17 | 0.24 | 63.1% |
| | 1200 | 0.04 | 0.25 | 63.2% |

In Table 12, we provide more fine-tuning experiment results on both *Bert-base-cased*(66) and *google/vit-base-patch16-224-in21k*(67) using the official example script from the "transformers" github repository(68). Under these two commonly used models, `FM-Delta` still shows good compression results. In fact, the small delta is caused by the fine-tuning itself. The pre-trained model parameters already have strong inference capabilities, so fine-tuning does not make much difference.

### E.5 Compression on Atypical models

To confirm the robustness of `FM-Delta`, we download two user-uploaded GPT-2 models from Huggingface: "vicgalle/gpt2-alpaca-gpt4' is a popular instruction-tuned model with high quality, and "jacksee/gpt2-finetuned-biochemistry' is an unpopular low-qualified model without any more information. Besides, we generate several atypical models by adding random tensors with noise values from 0 to {1, 10, 100, 1000} to the GPT-2 pre-trained model.

Table 13: Compression Rates and Perplexity on the popular, unpopular, and atypical models.

| Model | Comp. rate of FM-Delta | WikiText2 (ppl↓) |
| --- | --- | --- |
| vicgalle/gpt2-alpaca-gpt4 | 65.4% | 44.7223 |
| jacksee/gpt2-finetuned-biochemistry | 59.4% | 24564191.2047 |
| gpt2 with noise (0,1) | 85.3% | NaN |
| gpt2 with noise (0,10) | 87.0% | NaN |
| gpt2 with noise (0,100) | 92.2% | NaN |
| gpt2 with noise (0,1000) | 91.5% | NaN |

It can be seen in Table 13 that even on the GPT-2 added with huge noise (0,1000), `FM-Delta` can still compress the model into 91.5% of the original. This is consistent with our robustness analysis in Section 4, which illustrates that `FM-Delta` can accommodate a vast range of difference. Thus, `FM-Delta` is robust and reliable enough across diverse massive models in the cloud hub.

### E.6 Extended Time Results

Figure 13 presents the detailed time for model upload and download under different user bandwidths on <EleutherAI/gpt-neox-20b(52), KoboldAI/GPT-NeoX-20B-Erebus(69)>. When the user's bandwidth is below approximately 800Mbps, the total time is nearly equivalent to that of the non-compression solution for `FM-Delta`, and it is significantly reduced for `FM-Delta`$^U$ due to the decreased data transfer volume. When the user's bandwidth exceeds around 800Mbps, the total time is limited by the compression throughput due to the transmission speed being faster than the compression speed (approximately 100MB/s).

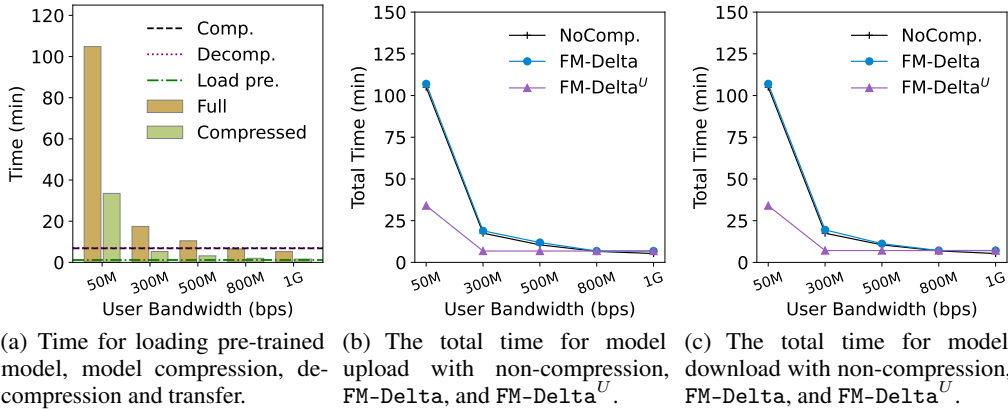

(a) Time for loading pre-trained model, model compression, decompression and transfer.

(b) The total time for model upload with non-compression, `FM-Delta`, and `FM-Delta`$^U$.

(c) The total time for model download with non-compression, `FM-Delta`, and `FM-Delta`$^U$.

Figure 13: End-to-end time under different user bandwidths on GPT-NeoX-20B.

### E.7 Cost Analysis

All the experiment results show that `FM-Delta` can achieve a compression rate around 50% in most cases with around 100MB/s throughput. There is an interesting question raising — in practice for the cloud, *what would be the cost of decompressing the models and would it be less than the cost of storing them decompressed in the cloud?* We mainly discuss the cloud cost in terms of storage and computation.

We assume that in cloud there is a model with size $M$, having $n$ fine-tuned variants. And $p$ of those fine-tuned variants are inactive (monthly downloads<10). We give a loose definition that requests received in the same minute we treat as concurrent requests. Assuming that the 10 downloads of inactive models are not concurrent, the probability of a single inactive model being downloaded in a certain minute is $\frac{10}{30*24*60}$. $k$ represents the number of concurrent download requests in a given minute and each request corresponds to a different fine-tuned model. Then we can think of model download as a binomial distribution:

$$P(X = k) = C_n^k \cdot \left( \frac{10}{30 \times 24 \times 60} \right)^k \cdot \left( 1 - \frac{10}{30 \times 24 \times 60} \right)^{n-k}. \tag{12}$$

Our goal is to determine the maximum total number $n$ of loadable models on the current server, which ensures that the probability of the concurrency number exceeding the server threshold $t$ is less than 1%, i.e., $\sum_{k=t}^{n} P(X = k) \leq 0.01$. Referring to Table 7, we set the threshold to 16, since its total time consumption is no more than twice that of a single task. We utilize python "scipy" library to compute and get the maximum value $n_{\max} = 35300$.

We refers to Table 1 to set the inactive proportion at $p = 89\%$. The model which can be compressed with FM-Delta in a minute is about 6GB, so we set the model storage size $4M = 6GB$. If 1TB costs 60 dollars, we give the original storage cost on inactive models without compression:

$$c_{\text{origin}} = 6/1024 * 35300 * 89\% * \$60 = \$11044.$$

With the compression rate $50\%$ of FM-Delta, the current storage cost is

$$c_{\text{fmdelta}} = 6/1024 * 35300 * 89\% * 50\% * \$60 = \$5522.$$

In addition, we directly present the purchase price of the current computing server to give the reader a more intuitive sense, even though the cloud server has a lot of computing power and we assume that there are idle computing resources. Our current server mainly consists of the cpu (one AMD Ryzen 9 5950X) and memory bars (Kingston Fury Beast 32GB×8), whose price in Amazon are $379 and 8×$100 respectively. Therefore, the purchase cost $c_{\text{server}}$ of the server is

$$c_{\text{server}} = 379 + 8 * 100 = \$1179.$$

From the above analysis we can conclude that the total cost of the cloud server is at least saved $11044 - 5522 - 1179 = 4343$ dollars on 35,300 inactive 6GB models. The example shows the storage costs are down to 57%. Such cost savings through lossless compression is certainly promising for the cloud in the real world.

